# Analog VLSI Circuits for Attention-Based, Visual Tracking

**Timothy K. Horiuchi**
Computation and Neural Systems
California Institute of Technology
Pasadena, CA 91125
timmer@klab.caltech.edu

**Tonia G. Morris**
Electrical and Computer Engineering
Georgia Institute of Technology
Atlanta, GA, 30332-0250
tmorris@eecom.gatech.edu

**Christof Koch**
Computation and Neural Systems
California Institute of Technology
Pasadena, CA 91125

**Stephen P. DeWeerth**
Electrical and Computer Engineering
Georgia Institute of Technology
Atlanta, GA, 30332-0250

## Abstract

A one-dimensional visual tracking chip has been implemented using neuromorphic, analog VLSI techniques to model selective visual attention in the control of saccadic and smooth pursuit eye movements. The chip incorporates focal-plane processing to compute image saliency and a winner-take-all circuit to select a feature for tracking. The target position and direction of motion are reported as the target moves across the array. We demonstrate its functionality in a closed-loop system which performs saccadic and smooth pursuit tracking movements using a one-dimensional mechanical eye.

## 1 Introduction

Tracking a moving object on a cluttered background is a difficult task. When more than one target is in the field of view, a decision must be made to determine which target to track and what its movement characteristics are. If motion information is being computed in parallel across the visual field, as is believed to occur in the middle temporal area (MT) of primates, some mechanism must exist to preferentially extract the activity of the neurons associated with the target at the appropriate

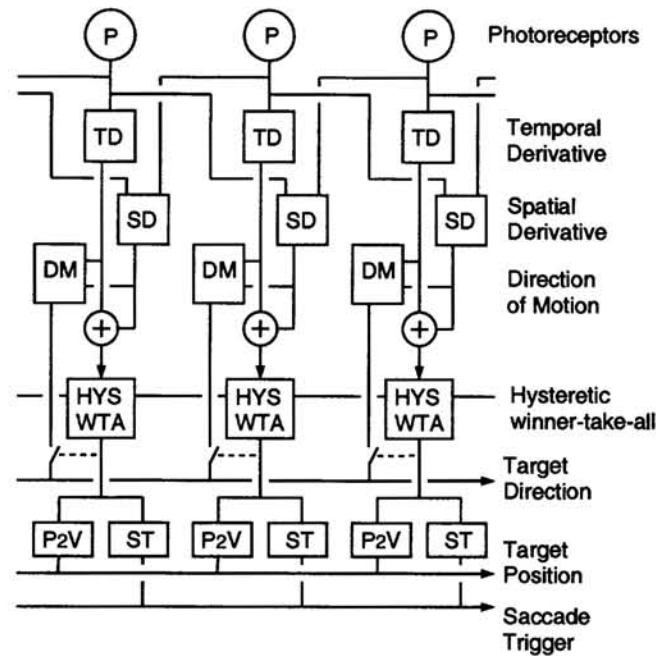

Figure 1: System Block Diagram: P = adaptive photoreceptor circuit, TD = temporal derivative circuit, SD = spatial derivative, DM = direction of motion, HYS WTA = hysteretic winner-take-all, P2V = position to voltage, ST = saccade trigger. The TD and SD are summed to form the saliency map from which the WTA finds the maximum. The output of the WTA steers the direction-of-motion information onto a common output line. Saccades are triggered when the selected pixel is outside a specified window located at the center of the array.

time. Selective visual attention is believed to be this mechanism.

In recent years, many studies have indicated that selective visual attention is involved in the generation of saccadic [10] [7] [12] [15] and smooth pursuit eye movements [9] [6] [16]. These studies have shown that attentional enhancement occurs at the target location just before a saccade as well as at the target location during smooth pursuit. In the case of saccades, attempts to dissociate attention from the target location has been shown to disrupt the accuracy or latency.

Koch and Ullman [11] have proposed a model for attentional selection based on the formation of a saliency map by combining the activity of elementary feature maps in a topographic manner. The most salient locations are where activity from many different feature maps coincide or at locations where activity from a preferentially-weighted feature map, such as temporal change, occurs. A winner-take-all (WTA) mechanism, acting as the center of the attentional "spotlight," selects the location with the highest saliency.

Previous work on analog VLSI-based, neuromorphic, hardware simulation of visual tracking include a one-dimensional, saccadic eye movement system triggered by temporal change [8] and a two-dimensional, smooth pursuit system driven by visual motion detectors [5]. Neither system has a mechanism for figure-ground discrimination of the target. In addition to this overt form of attentional shifting, covert

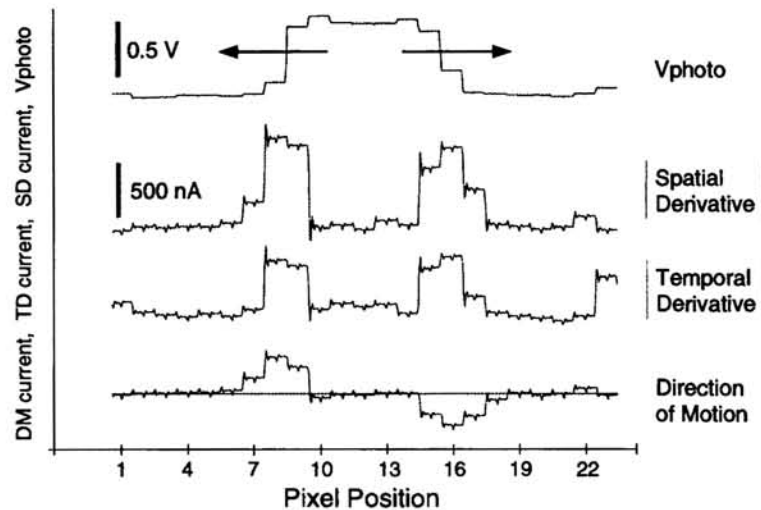

Figure 2: Example stimulus - Traces from top to bottom: Photoreceptor voltage, absolute value of the spatial derivative, absolute value of the temporal derivative, and direction-of-motion. The stimulus is a high-contrast, expanding bar, which provides two edges moving in opposite directions. The signed, temporal and spatial derivative signals are used to compute the direction-of-motion shown in the bottom trace.

attentional shifts have been modeled using analog VLSI circuits [4] [14], based on the Koch and Ullman model. These circuits demonstrate the use of delayed, transient inhibition at the selected location to model covert attentional scanning. In this paper we describe an analog VLSI implementation of an attention-based, visual tracking architecture which combines much of this previous work. Using a hardware model of the primate oculomotor system [8], we then demonstrate the use of the tracking chip for both saccadic and smooth pursuit eye movements.

## 2  System Description

The computational goal of this chip is the selection of a target, based on a given measure of saliency, and the extraction of its retinal position and direction of motion. Figure 1 shows a block diagram of the computation. The first few stages of processing compute simple feature maps which drive the WTA-based selection of a target to track. The circuits at the selected location signal their position and the computed direction-of-motion. This information is used by an external saccadic and smooth pursuit eye movement system to drive the eye. The saccadic system uses the position information to foveate the target and the smooth pursuit system uses the motion information to match the speed of the target.

Adaptive photoreceptors [2] (at the top of Figure 1) transduce the incoming pattern of light into an array of voltages. The temporal (TD) and spatial (SD) derivatives are computed from these voltages and are used to generate the saliency map and direction of motion. Figure 2 shows an example stimulus and the computed features. The saliency map is formed by summing the absolute-value of each derivative ($|TD| + |SD|$) and the direction-of-motion (DM) signal is a normalized product of the two

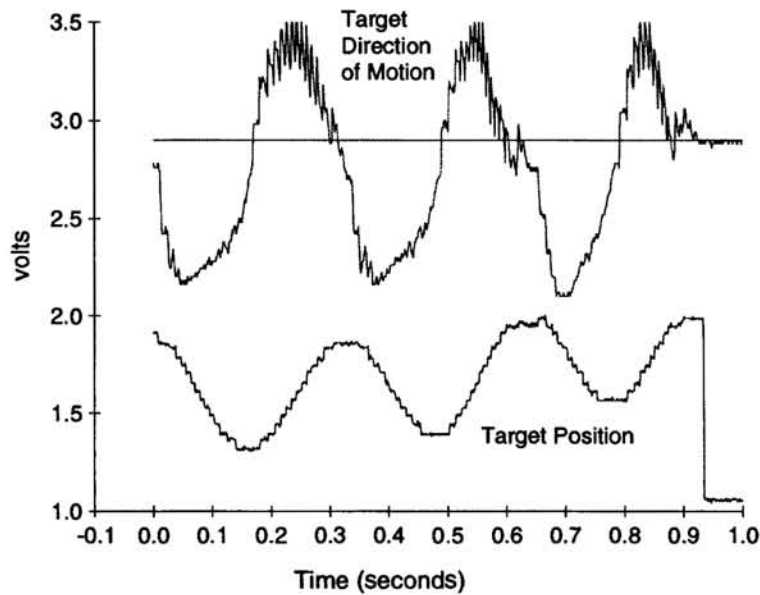

Figure 3: Extracting the target's direction of motion: The WTA output voltage is used to switch the DM current onto a common current sensing line. The output of this signal is seen in the top trace. The zero-motion level is indicated by the flat line shown at 2.9 volts. The lower trace shows the target's position from the position-to-voltage encoding circuits. The target's position and direction of motion are used to drive saccades and smooth pursuit eye movements during tracking.

derivatives. $\frac{TD \cdot SD}{|TD|+|SD|}$

In the saliency map, the temporal and spatial derivatives can be differentially weighted to emphasize moving targets over stationary targets. The saliency map provides the input to a winner-take-all (WTA) computation which finds the maximum in this map. Spatially-distributed hysteresis is incorporated in this winner-take-all computation [4] by adding a fixed current to the winner's input node and its neighbors. This distributed hysteresis is motivated by the following two ideas: 1) once a target has been selected it should continue to be tracked even if another equally interesting target comes along, and 2) targets will typically move continuously across the array. Hysteresis reduces oscillation of the winning status in the case where two or more inputs are very close to the winning input level and the local distribution of hysteresis allows the winning status to freely shift to neighboring pixels rather than to another location further away.

The WTA output signal is used to drive three different circuits: the position-to-voltage (P2V) circuit [3], the DM-current-steering circuit (see Figure 3), and the saccadic triggering (ST) circuit. The only circuits that are active are those at the winning pixel locations. The P2V circuit drives the common position output line to a voltage representing it's position in the array, the DM-steering circuit puts the local DM circuit's current onto the common motion output line, and the ST circuit drives a position-specific current onto a common line to be compared against an externally-set threshold value. By creating a "V" shaped profile of ST currents centered on the array, winning pixels away from the center will exceed the threshold

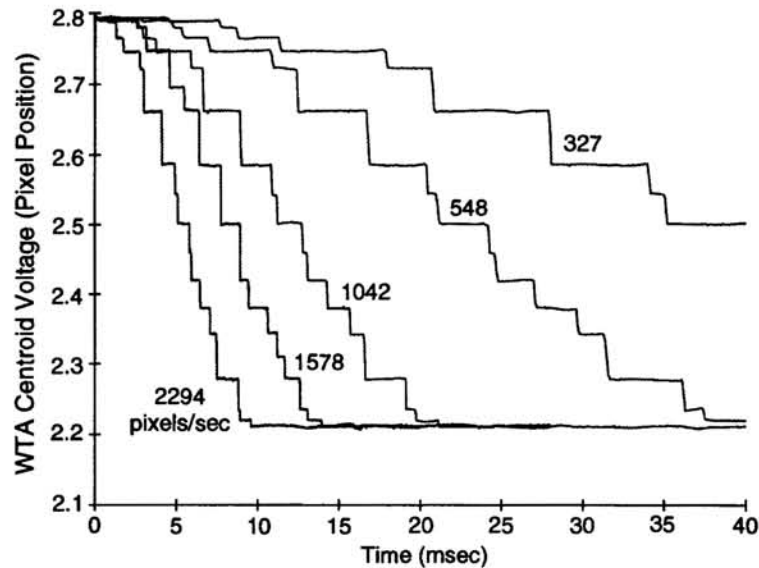

Figure 4: Position vs. time traces for the passage of a strong edge across the array at five different speeds. The speeds shown correspond to 327, 548, 1042, 1578, 2294 pixels/sec.

and send saccade requests off-chip. Figure 3 shows the DM and P2V outputs for an oscillating target.

To test the speed of the tracking circuit, a single edge was passed in front of the array at varying speeds. Figure 4 shows some of these results. The power consumption of the chip (23 pixels and support circuits, not including the pads) varies between 0.35 mW and 0.60 mW at a supply voltage of 5 volts. This measurement was taken with no clock signal driving the scanners since this is not essential to the operation of the circuit.

# 3   System Integration

The tracking chip has been mounted on a neuromorphic, hardware model of the primate oculomotor system [8] and is being used to track moving visual targets. The visual target is mounted to a swinging apparatus to generate an oscillating motion. Figure 5 shows the behavior of the system when the retinal target position is used to drive re-centering saccades and the target direction of motion is used drive smooth pursuit. Saccades are triggered when the selected pixel is outside a specified window centered on the array and the input to the smooth pursuit system is suppressed during saccades. The smooth pursuit system mathematically integrates retinal motion to match the eye velocity to the target velocity.

# 4   Acknowledgements

T. H. is supported by an Office of Naval Research AASERT grant and by the NSF Center for Neuromorphic Systems Engineering at Caltech. T. M. is supported by the Georgia Tech Research Institute.

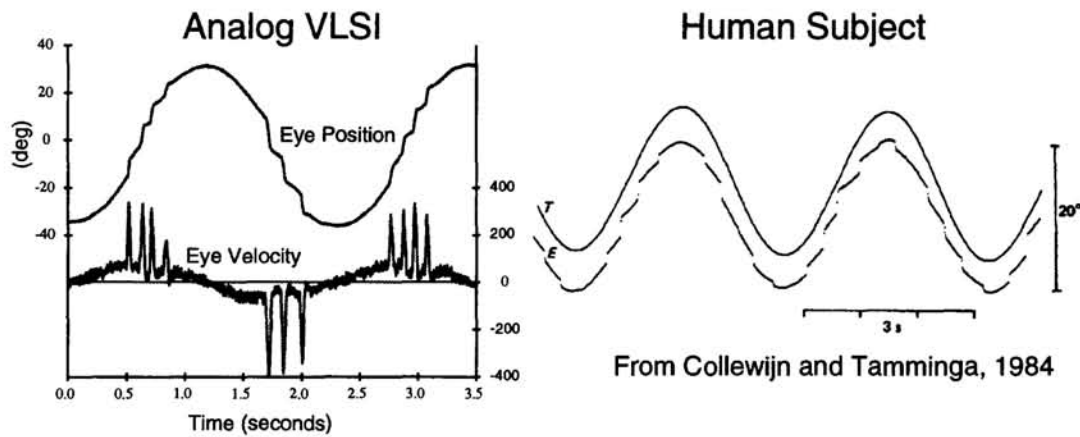

Figure 5: Saccades and Smooth Pursuit: In this example, a swinging target is tracked over a few cycles. Re-centering saccades are triggered when the target leaves a specified window centered on the array. For comparison, on the right, we show human data for the same task [1].

# References

[1] H. Collewijn and E. Tamminga, "Human smooth and saccadic eye movements during voluntary pursuit of different target motions on different backgrounds" *J. Physiol.*, Vol. 351, pp. 217-250. (1984)

[2] T. Delbrück, Ph.D. Thesis, Computation and Neural Systems Program California Institute of Technology (1993)

[3] S. P. DeWeerth, "Analog VLSI Circuits for Stimulus Localization and Centroid Computation" *Intl. J. Comp. Vis.* 8(3), pp. 191-202. (1992)

[4] S. P. DeWeerth and T. G. Morris, "CMOS Current Mode Winner-Take-All with Distributed Hysteresis" *Electronics Letters*, Vol. 31, No. 13, pp. 1051-1053. (1995)

[5] R. Etienne-Cummings, J. Van der Spiegel, and P. Mueller "A Visual Smooth Pursuit Tracking Chip" *Advances in Neural Information Processing Systems 8* (1996)

[6] V. Ferrara and S. Lisberger, "Attention and Target Selection for Smooth Pursuit Eye Movements" *J. Neurosci.*, 15(11), pp. 7472-7484, (1995)

[7] J. Hoffman and B. Subramaniam, "The Role of Visual Attention in Saccadic Eye Movements" *Perception and Psychophysics*, 57(6), pp. 787-795, (1995)

[8] T. Horiuchi, B. Bishofberger, and C. Koch, "An Analog VLSI Saccadic System" *Advances in Neural Information Processing Systems 6*, Morgan Kaufmann, pp. 582-589, (1994)

[9] B. Khurana, and E. Kowler, "Shared Attentional Control of Smooth Eye Movement and Perception" *Vision Research*, 27(9), pp. 1603-1618, (1987)

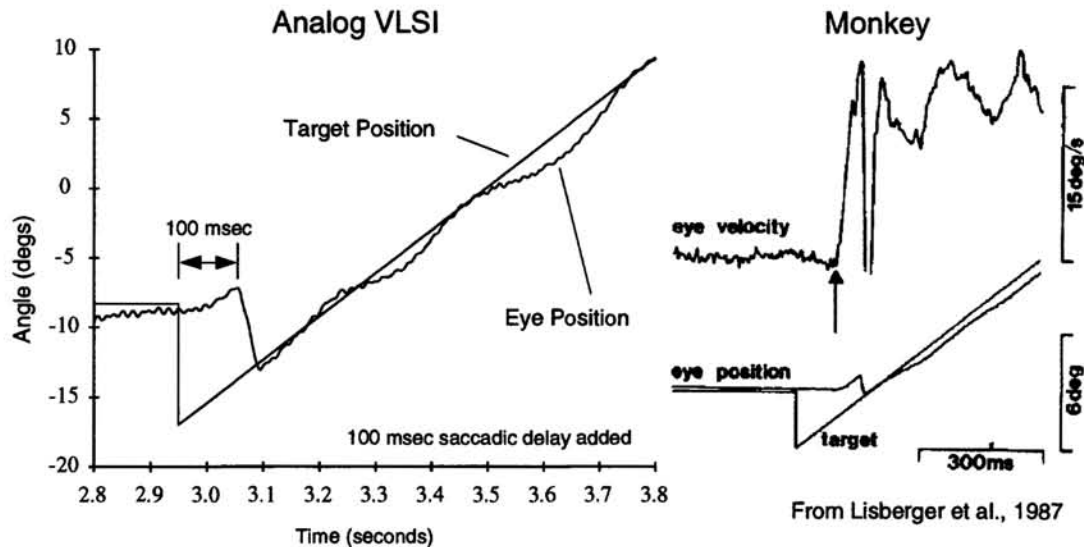

Figure 6: Step-Ramp Experiment: In this experiment, the target jumps from the fixation point to a new location and begins moving with constant velocity. On the left, the analog VLSI system tracks the target. For comparison, on the right, we show data from a monkey performing the same task [13].

[10] E. Kowler, E. Anderson, B. Dosher, E. Blaser, "The Role of Attention in the Programming of Saccades" *Vision Research*, 35(13), pp. 1897-1916, (1995)

[11] C. Koch and S. Ullman, "Shifts in selective visual attention: towards the underlying neural circuitry" *Human Neurobiology*, 4:219-227, (1985)

[12] R. Rafal, P. Calabresi, C. Brennan, and T. Scioltio, "Saccade Preparation Inhibits Reorienting to Recently Attended Locations" *J. Exp. Psych: Hum. Percep. and Perf.*, 15, pp. 673-685, (1989)

[13] S. G. Lisberger, E. J. Morris, and L. Tychsen, "Visual motion processing and sensory-motor integration for smooth pursuit eye movements." In *Ann. Rev. Neurosci.*, Cowan et al., editors. Vol. 10, pp. 97-129, (1987)

[14] T. G. Morris and S. P. DeWeerth, "Analog VLSI Circuits for Covert Attentional Shifts" *Proc. 5th Intl. Conf. on Microelectronics for Neural Networks and Fuzzy Systems* - MicroNeuro96, Feb 12-14, 1996. Lausanne, Switzerland, IEEE Computer Society Press, Los Alamitos, CA, pp. 30-37, (1996)

[15] S. Shimojo, Y. Tanaka, O. Hikosaka, and S. Miyauchi, "Vision, Attention, and Action – inhibition and facilitation in sensory motor links revealed by the reaction time and the line-motion." In *Attention and Performance XVI*, T. Inui & J. L. McClelland, editors. MIT Press, (1995)

[16] W. J. Tam and H. Ono, "Fixation Disengagement and Eye-Movement Latency" *Perception and Psychophysics*, 56(3) pp. 251-260, (1994)
